# Two-layer Generalization Analysis for Ranking Using Rademacher Average

**Wei Chen**[*]
Chinese Academy of Sciences
chenwei@amss.ac.cn

**Tie-Yan Liu**
Microsoft Research Asia
tyliu@micorsoft.com

**Zhiming Ma**
Chinese Academy of Sciences
mazm@amt.ac.cn

## Abstract

This paper is concerned with the generalization analysis on learning to rank for information retrieval (IR). In IR, data are hierarchically organized, i.e., consisting of queries and documents. Previous generalization analysis for ranking, however, has not fully considered this structure, and cannot explain how the simultaneous change of query number and document number in the training data will affect the performance of the learned ranking model. In this paper, we propose performing generalization analysis under the assumption of two-layer sampling, i.e., the i.i.d. sampling of queries and the conditional i.i.d sampling of documents per query. Such a sampling can better describe the generation mechanism of real data, and the corresponding generalization analysis can better explain the real behaviors of learning to rank algorithms. However, it is challenging to perform such analysis, because the documents associated with different queries are not identically distributed, and the documents associated with the same query become no longer independent after represented by features extracted from query-document matching. To tackle the challenge, we decompose the expected risk according to the two layers, and make use of the new concept of two-layer Rademacher average. The generalization bounds we obtained are quite intuitive and are in accordance with previous empirical studies on the performances of ranking algorithms.

## 1 Introduction

Learning to rank has recently gained much attention in machine learning, due to its wide applications in real problems such as information retrieval (IR). When applied to IR, learning to rank is a process as follows [16]. First, a set of queries, their associated documents, and the corresponding relevance judgments are given. Each document is represented by a set of features, measuring the matching between document and query. Widely-used features include the frequency of query terms in the document and the query likelihood given by the language model of the document. A ranking function, which combines the features to predict the relevance of a document to the query, is learned by minimizing a loss function defined on the training data. Then for a new query, the ranking function is used to rank its associated documents according to their predicted relevance. Many learning to rank algorithms have been proposed, among which the pairwise ranking algorithms such as Ranking SVMs [12, 13], RankBoost [11], and RankNet [5] have been widely applied.

To understand existing ranking algorithms, and to guide the development of new ones, people have studied the learning theory for ranking, in particular, the generalization ability of ranking methods. Generalization ability is usually represented by a bound of the deviation between the expected and empirical risks for an arbitrary ranking function in the hypothesis space. People have investigated the generalization bounds under different assumptions. First, with the assumption that documents are i.i.d., the generalization bounds of RankBoost [11], stable pairwise ranking algorithms like Ranking

---

[*]The work was performed when the first author was an intern at Microsoft Research Asia.

SVMs [2], and algorithms minimizing pairwise 0-1 loss [1, 9] were studied. We call these generalization bounds "document-level generalization bounds", which converge to zero when the number of documents in the training set approaches infinity. Second, with the assumption that queries are i.i.d., the generalization bounds of stable pairwise ranking algorithms like Ranking SVMs and IR-SVM [6] and listwise algorithms were obtained in [15] and [14]. We call these generalization bounds "query-level generalization bounds". When analyzing the query-level generalization bounds, the documents associated with each query are usually regarded as a deterministic set [10, 14], and no random sampling of documents is assumed. As a result, query-level generalization bounds converge to zero only when the number of queries approaches infinity, no matter how many documents are associated with them.

While the existing generalization bounds can explain the behaviors of some ranking algorithms, they also have their limitations. (1) The assumption that documents are i.i.d. makes the document-level generalization bounds not directly applicable to ranking in IR. This is because it has been widely accepted that the documents associated with different queries do not follow the same distribution [17] and the documents with the same query are no longer independent after represented by document-query matching features. (2) It is not reasonable for query-level generalization bounds to assume that one can obtain the document set associated with each query in a deterministic manner. Usually there are many random factors that affect the collection of documents. For example, in the labeling process of TREC, the ranking results submitted by all TREC participants were put together and then a proportion of them were selected and presented to human annotators for labeling. In this process, the number of participants, the ranking result given by each participant, the overlap between different ranking results, the labeling budget, and the selection methodology can all influence which documents and how many documents are labeled for each query. As a result, it is more reasonable to assume a random sampling process for the generation of labeled documents per query.

To address the limitations of previous work, we propose a novel theoretical framework for ranking, in which a two-layer sampling strategy is assumed. In the first layer, queries are i.i.d. sampled from the query space according to a fixed but unknown probability distribution. In the second layer, for each query, documents are i.i.d. sampled from the document space according to a fixed but unknown conditional probability distribution determined by the query (i.e., documents associated with different queries do not have the identical distribution). Then, a set of features are extracted for each document with respect to the query. Note that the feature representations of the documents with the same query, as random variables, are not independent any longer. But they are conditionally independent if the query is given. As can be seen, this new sampling strategy removes improper assumptions in previous work, and can more accurately describe the data generation process in IR.

Based on the framework, we have performed two-layer generalization analysis for pairwise ranking algorithms. However, the task is non-trivial mainly because the two-layer sampling does not correspond to a typical empirical process: the documents for different queries are not identically distributed while the documents for the same query are not independent. Thus, the empirical process techniques, widely used in previous work on generalization analysis, are not sufficient. To tackle the challenge, we carefully decompose the expected risk according to the query and document layers, and employ a new concept called two-layer Rademacher average. The new concept accurately describes the complexity in the two-layer setting, and its reduced versions can be used to derive meaningful bounds for query layer error and document layer error respectively.

According to the generalization bounds we obtained, we have the following findings: (i) Both more queries and more documents per query can enhance the generalization ability of ranking methods; (ii) Only if both the number of training queries and that of documents per query simultaneously approach infinity, can the generalization bound converge to zero; (iii) Given a fixed size of training data, there exists an optimal tradeoff between the number of queries and the number of documents per query. These findings are quite intuitive and can well explain empirical observations [19].

## 2  Related Work

### 2.1  Pairwise Learning to Rank

Pairwise ranking is one of the major approaches to learning to rank, and has been widely adopted in real applications [5, 11, 12, 13]. The process of pairwise ranking can be described as follows.

Assume there are $n$ queries $\{q_1, q_2, \cdots, q_n\}$ in the training data. Each query $q_i$ is associated with $m_i$ documents $\{d_1^i, \cdots, d_{m_i}^i\}$ and their judgments $\{y_1^i, \cdots, y_{m_i}^i\}$, where $y_j^i \in \mathcal{Y}$. Each document $d_j^i$ is represented by a set of features $x_j^i = \psi(d_j^i, q_i) \in \mathcal{X}$, measuring the matching between document $d_j^i$ and query $q_i$. Widely-used features include the frequency of query terms in the document and the query likelihood given by the language model of the document. For ease of reference, we use $z = (x, y) \in \mathcal{X} \times \mathcal{Y} = \mathcal{Z}$ to denote document $d$ since it encodes all the information of $d$ in the learning process. Then the training set can be denoted as $S = \{S_1, \cdots, S_n\}$ where $S_i \triangleq \{z_j^i \in \mathcal{Z}\}_{j=1,\cdots,m_i}$ is the document sample for query $q_i$. For a ranking function $f : \mathcal{X} \to R$, the pairwise 0-1 loss $l_{0-1}$ and pairwise surrogate loss $l_\phi$ are defined as below:

$$l_{0-1}(z, z'; f) = I_{\{(y-y')(f(x)-f(x'))<0\}},$$
$$l_\phi(z, z'; f) = \phi\big(-sgn(y - y') \cdot (f(x) - f(x'))\big), \tag{1}$$

where $I_{\{\cdot\}}$ is the indicator function and $z, z'$ are two documents associated with the same query. When function $\phi$ takes different forms, we will get the surrogate loss functions for different algorithms. For example, for Ranking SVMs, RankBoost, and RankNet, function $\phi$ is the hinge, exponential, and logistic functions respectively.

## 2.2 Document-level Generalization Analysis

In document-level generalization analysis, it is assumed that the documents are i.i.d. sampled from the document space $\mathcal{Z}$ according to $P(z)$. Then the expected risk of pairwise ranking algorithms can be defined as below,

$$R_D^l(f) = \int_{\mathcal{Z}^2} l(z, z'; f) dP^2(z, z'),$$

where $P^2(z, z')$ is the product probability of $P(z)$ on the product space $\mathcal{Z}^2$.

The document-level generalization bound usually takes the following form: with probability at least $1 - \delta$,

$$R_D^l(f) \le \frac{1}{m(m-1)} \sum_{j \ne k} l(z_j, z_k; f) + \epsilon(\delta, \mathcal{F}, m), \forall f \in \mathcal{F},$$

where $\epsilon(\delta, \mathcal{F}, m) \to 0$ when document number $m \to \infty$.

As representative work, the generalization bounds for the pairwise 0-1 loss were derived in [1, 9] and the generalization bounds for RankBoost and Ranking SVMs were obtained in [2] and [11].

As aforementioned, the assumption that documents are i.i.d. makes the document-level generalization bounds not directly applicable to ranking in IR. Even if the assumption holds, the document-level generalization bounds still cannot be used to explain existing pairwise ranking algorithms. Actually, according to the document-level generalization bound, what we can obtain is: with probability at least $1 - \delta$, $R_D^l(f) \le \frac{\sum_{(i,j) \ne (i',k)} l(z_j^i, z_k^{i'}; f)}{\sum m_i (\sum m_i - 1)} + \epsilon(\delta, \mathcal{F}, \sum m_i), \forall f \in \mathcal{F}$. The empirical risk is the average of the pairwise losses on all the document pairs. This is clearly not the real empirical risk of ranking in IR, where documents associated with different queries cannot be compared with each other, and pairs are constructed only by documents associated with the same query.

## 2.3 Query-level Generalization Analysis

In existing query-level generalization analysis [14], it is assumed that each query $q_i$, represented by a deterministic document set $S_i$ with the same number of documents (i.e. $m_i \equiv m$), is i.i.d. sampled from the space $\mathcal{Z}^m$. Then the expected risk can be defined as follows,

$$R_Q^l(f) = \int_{\mathcal{Z}^m} \frac{1}{m(m-1)} \sum_{j \ne k} l(z_j, z_k; f) dP(z_1, \cdots, z_m).$$

The query-level generalization bound usually takes the following form: with probability at least $1 - \delta$,

$$R_Q^l(f) \le \frac{1}{n} \sum_{i=1}^n \frac{1}{m(m-1)} \sum_{j \ne k} l(z_j^i, z_k^i; f) + \epsilon(\delta, \mathcal{F}, n), \forall f \in \mathcal{F},$$

where $\epsilon(\delta, \mathcal{F}, n) \to 0$ as query number $n \to \infty$.

As representative work, the query-level generalization bounds for stable pairwise ranking algorithms such as Ranking SVMs and IR-SVM and listwise ranking algorithms were derived in [15] [1] and [14].

As mentioned in the introduction, the assumption that each query is associated with a deterministic set of documents is not reasonable. The fact is that many random factors can influence what kinds of documents and how many documents are labeled for each query. Due to this inappropriate assumption, the query-level generalization bounds are sometimes not intuitive. For example, when more labeled documents are added to the training set, the generalization bounds of stable pairwise ranking algorithms derived in [15] do not change and the generalization bounds of some of the listwise ranking algorithms derived in [14] get even looser.

## 3  Two-Layer Generalization Analysis

In this section, we introduce the concepts of two-layer data sampling and two-layer generalization ability for ranking. These concepts can help describe the data generation process and explain the behaviors of learning to rank algorithms more accurately than previous work.

### 3.1  Two-Layer Sampling in IR

When applying learning to rank techniques to IR, a training set is needed. The creation of such a training set is usually as follows. First, queries are randomly sampled from query logs of search engines. Then for each query, documents that are potentially relevant to the query are sampled (e.g., using the strategy in TREC [8]) from the entire document repository and presented to human annotators. Human annotators make relevance judgment to these documents, according to the matching between them and the query. Mathematically, we can represent the above process in the following manner. First, queries $Q = \{q_1, \cdots, q_n\}$ are i.i.d. sampled from the query space $\mathcal{Q}$ according to distribution $P(q)$. Second, for each query $q_i$, its associated documents and their relevant judgments $\{(d_1^i, y_1^i), \cdots, (d_{m_i}^i, y_{m_i}^i)\}$ are i.i.d. sampled from the document space $\mathcal{D}$ according to a conditional distribution $P(d|q_i)$ where $m_i$ is the number of sampled documents. Each document $d_j^i$ is then represented by a set of matching features, i.e., $x_j^i = \psi(d_j^i, q_i)$, where $\psi$ is a feature extractor. Following the notation rules in Section 2.1, we use $z_j^i = (x_j^i, y_j^i)$ to represent document $d_j^i$ and its label, and denote the training data for query $q_i$ as $S_i = \{z_j^i\}_{j=1,\cdots,m_i}$. Note that although $\{d_j^i\}_{j=1,\cdots,m_i}$ are i.i.d. samples, random variables $\{z_j^i\}_{j=1,\cdots,m_i}$ are no longer independent because they share the same query $q_i$. Only if $q_i$ is given, we can regard them as independent of each other.

We call the above data generation process *two-layer sampling*, and denote the training data generated in this way as $(Q, S)$, where $Q$ is the query sample and $S = \{S_i\}_{i=1,\cdots,n}$ is the document sample. The two-layer sampling process can be illustrated using Figure 1.

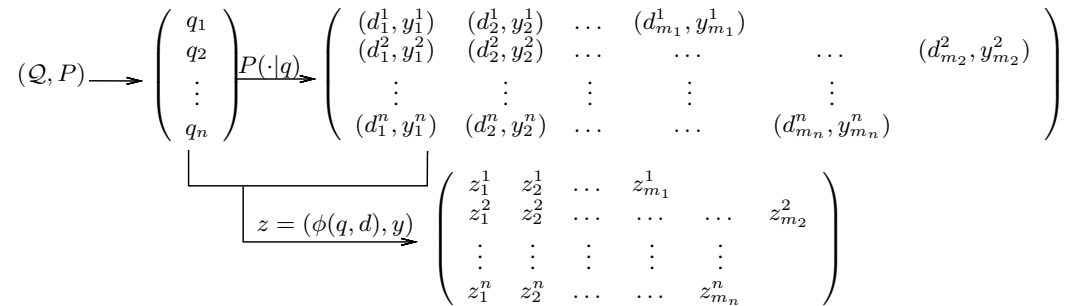

Figure 1: Two-layer sampling

Note that two-layer sampling has significant difference from the sampling strategies used in previous generalization analysis. (i) As compared to the sampling in document-level generalization analysis, two-layer sampling introduces the sampling of queries, and documents associated with

$$(\mathcal{P}, P) \quad \rightarrow \quad \begin{matrix} P_1 & \rightarrow \\ P_2 & \rightarrow \\ \vdots & \vdots \\ P_n & \rightarrow \end{matrix} \quad \begin{pmatrix} z_1^1 & z_2^1 & \dots & z_m^1 \\ z_1^2 & z_2^2 & \dots & z_m^2 \\ \vdots & \vdots & \vdots & \vdots \\ z_1^n & z_2^n & \dots & z_m^n \end{pmatrix}$$

Figure 2: $(n, m)$-sampling

different queries are sampled according to different conditional distributions. (ii) As compared to the sampling in query-level generalization analysis, two-layer sampling considers the sampling of documents for each query.

To some extent, the aforementioned two-layer sampling has relationship with directly sampling from the product space of query and document, and the $(n, m)$-sampling proposed in [4]. However, as shown below, they also have significant differences. Firstly, it is clear that directly sampling from the product space of query and document does not describe the real data generation process. Furthermore, even if we sample a large number of documents in this way, it is not guaranteed that we can have sufficient number of documents for each single query. Secondly, comparing Figure 1 with Figure 2 (which illustrate $(n, m)$-sampling), we can easily find: (i) in $(n, m)$-sampling, tasks (corresponding to queries) have the same number (i.e., $m$) of elements (corresponding to documents), however, in two-layer sampling, queries can be associated with different numbers of documents; (ii) in $(n, m)$-sampling, all the elements are i.i.d., however, in two-layer sampling documents (if represented by matching features) associated with the same query are not independent of each other.

### 3.2 Two-Layer Generalization Ability

With the probabilistic assumption of two-layer sampling, we define the expected risk for pairwise ranking as follows,

$$R^l(f) = \int_{\mathcal{Q}} \int_{\mathcal{Z}^2} l(z, z'; f) dP(z, z'|q) dP(q), \tag{2}$$

where $P(z, z'|q)$ is the product probability of $P(z|q)$ on the product space $\mathcal{Z}^2$.

**Definition 1.** *We say that an ERM learning process with loss $l$ in hypothesis space $\mathcal{F}$ has two-layer generalization ability, if with probability at least $1 - \delta$,*

$$R^l(f) \leq \hat{R}^l_{n;m_1,\cdots,m_n}(f; S) + \epsilon(\delta, \mathcal{F}, n, m_1, \cdots, m_n), \forall f \in \mathcal{F},$$

*where $\hat{R}^l_{n;m_1,\cdots,m_n}(f; S) = \frac{1}{n} \sum_{i=1}^n \frac{1}{m_i(m_i-1)} \sum_{j \neq k} l(z_j^i, z_k^i; f)$, and $\epsilon(\delta, \mathcal{F}, n, m_1, \cdots, m_n) \to 0$ iff query number $n$ and document number per query $m_i$ simultaneously approach infinity.*

In the next section, we will show our theoretical results on the two-layer generalization abilities of typical pairwise ranking algorithms.

## 4 Main Theoretical Result

In this section, we show our results on the two-layer generalization ability of ERM learning with pairwise ranking losses (either pairwise 0-1 loss or pairwise surrogate losses). As prerequisites, we recall the concept of conventional Rademacher averages (RA)[3].

**Definition 2.** *For sample $\{x_1, \ldots, x_m\}$, the RA of $l \circ \mathcal{F}$ is defined as follows, $\mathcal{R}_m(l \circ \mathcal{F}) = \mathbf{E}_\sigma \left[ \sup_{f \in \mathcal{F}} \left| \frac{2}{m} \sum_{j=1}^m \sigma_j l(x_j; f) \right| \right]$, where $\sigma_1, \ldots, \sigma_m$ are independent Rademacher random variables independent of data sample.*

With the above definitions, we have the following theorem, which describes when and how the two-layer generalization bounds of pairwise ranking algorithms converge to zero.

**Theorem 1.** *Suppose $l$ is the loss function for pairwise ranking. Assume 1) $l \circ \mathcal{F}$ is bounded by $M$, 2) $\mathbf{E}\left[\mathcal{R}_m(l \circ \mathcal{F})\right] \leq D(l \circ \mathcal{F}, m)$, then with probability at least $1 - \delta$, for $\forall f \in \mathcal{F}$*

$$R^l(f) \leq \hat{R}^l_{n;m_1,\cdots,m_n}(f) + D(l \circ \mathcal{F}, n) + \sqrt{\frac{2M^2 \log(\frac{4}{\delta})}{n}} + \frac{1}{n} \sum_{i=1}^n D(l \circ \mathcal{F}, \lfloor \frac{m_i}{2} \rfloor) + \sqrt{\sum_{i=1}^n \frac{2M^2 \log \frac{4}{\delta}}{m_i n^2}}.$$

**Remark:** The condition of the existence of upper bounds for $\mathbf{E}\left[\mathcal{R}_m(l \circ \mathcal{F})\right]$ can be satisfied in many situations. For example, for ranking function class $\mathcal{F}$ that satisfies $VC(\tilde{\mathcal{F}}) = V$, where $\tilde{\mathcal{F}} = \{f(x, x') = f(x) - f(x'); f \in \mathcal{F}\}$, $VC(\cdot)$ denotes the VC dimension, and $|f(x)| \leq B$, it has been proved that $D(l_{0-1} \circ \mathcal{F}, m) = c_1 \sqrt{V/m}$ and $D(l_\phi \circ \mathcal{F}, m) = c_2 B \phi'(B) \sqrt{V/m}$ in [3, 9], where $c_1$ and $c_2$ are both constants.

## 4.1 Proof of Theorem 1

Note that the proof of Theorem 1 is non-trivial because documents generated by two-layer sampling are neither independent nor identically distributed, as aforementioned. As a result, the two-layer sampling does not correspond to an empirical process and classical proving techniques in statistical learning are not sufficient for the proof. To tackle the challenge, we decompose the two-layer expected risk as follows:

$$R^l(f) = \hat{R}^l_{n;m_1,\cdots,m_n}(f) + R^l(f) - \hat{R}^l_n(f) + \hat{R}^l_n(f) - \hat{R}^l_{n;m_1,\cdots,m_n}(f),$$

where $\hat{R}^l_n(f) = \frac{1}{n} \sum_{i=1}^n \int_{\mathcal{Z}^2} l(z, z'; f) dP(z, z'|q_i)$. We call $R^l(f) - \hat{R}^l_n(f)$ *query-layer error* and $\hat{R}^l_n(f) - \hat{R}^l_{n;m_1,\cdots,m_n}(f)$ *document-layer error*. Then, inspired by conventional RA [3], we propose a concept called *two-layer RA* to describe the complexity of sample $(Q, S)$.

**Definition 3.** *For two-layer sample $(Q, S)$, the two-layer RA of $l \circ \mathcal{F}$ is defined as follows,*

$$\mathcal{R}_{n;m_1,\cdots,m_n}(l \circ \mathcal{F}(Q, S)) = \mathbf{E}_\sigma \left[ \sup_{f \in \mathcal{F}} \left| \frac{2}{n} \sum_{i=1}^n \frac{1}{\lfloor m_i/2 \rfloor} \sum_{j=1}^{\lfloor m_i/2 \rfloor} \sigma^i_j l(z^i_j, z^i_{\lfloor m_i/2 \rfloor + j}; f) \right| \right],$$

*where $\{\sigma^i_j\}$ are independent Rademacher random variables independent of data sample. If $(Q, S) = \{q_i; z_i, z'_i\}_{i=1,\cdots,n}$, we call its expected two-layer RA, i.e., $\mathbf{E}_{Q,S}\left[\mathcal{R}_{n;2,\cdots,2}(l \circ \mathcal{F}(Q, S))\right]$, document-layer reduced two-layer RA. If $(q, S) = \{q; z_1, \cdots, z_m\}$, we call its conditional expected two-layer RA, i.e., $\mathbf{E}_{S|q}\left[\mathcal{R}_{1;m}(l \circ \mathcal{F}(q, S))\right]$, query-layer reduced two-layer RA.*

Based on the concept of two-layer RA, we can derive meaningful bounds for the two-layer expected risk. In Section 4.1.1, we prove the query-layer error bound by using *document-layer reduced two-layer RA*; and in Section 4.1.2, we prove the document-layer error bound by using *query-layer reduced two-layer RA*. Combining the two bounds, we can prove Theorem 1 in Section 4.1.3.

### 4.1.1 Query-Layer Error Bounds

As for the query-layer error bound, we have the following theorem.

**Theorem 2.** *Assume $l \circ \mathcal{F}$ is bounded by $M$, then with probability at least $1 - \delta$,*

$$R^l(f) - \hat{R}^l_n(f) \leq \mathbf{E}_{Q,S}\left[\mathcal{R}_{n;2,\cdots,2}(l \circ \mathcal{F}(Q, S))\right] + \sqrt{\frac{2M^2 \log(2/\delta)}{n}}, \forall f \in \mathcal{F}.$$

*Proof.* We define a function $L_f$ as follows: $L_f(q) = \int_{\mathcal{Z}^2} l(z, z'; f) dP^2(z|q)$. Since $q_1, \cdots, q_n$ are i.i.d. sampled, $L_f(q_1), \cdots, L_f(q_n)$ are also i.i.d.. Denote $G_1(Q) = \sup_{f \in \mathcal{F}} \left| R^l(f) - \hat{R}^l_n(f) \right|$. Since $l \circ \mathcal{F}$ is bounded by $M$, by the McDiarmid's inequality, we have $G_1(Q) \leq \mathbf{E}\left[G_1(Q)\right] + \sqrt{\frac{2M^2 \log(\frac{2}{\delta})}{n}}$. By introducing a ghost query sample $\tilde{Q} = \{\tilde{q}_1, \cdots, \tilde{q}_n\}$, we have

$$\mathbf{E}\left[G_1(Q)\right] = \mathbf{E}_Q \left[ \sup_{f \in \mathcal{F}} \left| \frac{1}{n} \sum_{i=1}^n L_f(q_i) - \int L_f(q) dP(q) \right| \right] \leq \mathbf{E}_{Q,\tilde{Q}} \left[ \sup_{f \in \mathcal{F}} \left| \frac{1}{n} \sum_{i=1}^n L_f(q_i) - \frac{1}{n} \sum_{i=1}^n L_f(\tilde{q}_i) \right| \right] \quad (3)$$

Further assuming that there are virtual document samples $\{z_i, z'_i\}_{i=1,\cdots,n}$ and $\{\tilde{z}_i, \tilde{z'}_i\}_{i=1,\cdots,n}$ for query samples $Q$ and $\tilde{Q}$, we have $L_f(q_i) = \mathbf{E}_{z_i, z'_i|q_i}[l(z_i, z'_i; f)]; L_f(\tilde{q}_i) = \mathbf{E}_{\tilde{z}_i, \tilde{z'}_i|\tilde{q}_i}[l(\tilde{z}_i, \tilde{z'}_i; f)]$. Substitute $L_f(q_i)$ and $L_f(\tilde{q}_i)$ into inequality 3, we obtain the following result:

$$\mathbf{E}[G_1(Q)] = \mathbf{E}_{Q,Q'} \left[ \sup_{f \in \mathcal{F}} \left| \frac{1}{n} \sum_{i=1}^n \mathbf{E}_{z_i, z'_i, \tilde{z}_i, \tilde{z'}_i|q_i, \tilde{q}_i} \left( l(z_i, z'_i; f) - l(\tilde{z}_i, \tilde{z'}_i; f) \right) \right| \right]$$

$$\leq \mathbf{E}_{q_i, \tilde{q}_i} \mathbf{E}_{z_i, z'_i, \tilde{z}_i, \tilde{z'}_i|q_i, \tilde{q}_i} \left[ \sup_{f \in \mathcal{F}} \left| \frac{1}{n} \sum_{i=1}^n \left( l(z_i, z'_i; f) - l(\tilde{z}_i, \tilde{z'}_i; f) \right) \right| \right] = \mathbf{E}_{Q,S} \mathbf{E}_\sigma \left[ \sup_{f \in \mathcal{F}} \left| \frac{2}{n} \sum_{i=1}^n \sigma_i l(z_i, z'_i; f) \right| \right]$$

According to the definition of document-layer reduced two-layer RA, Theorem 2 is proved. $\quad\square$

### 4.1.2 Document-layer Error Bound

In order to obtain the bound for document-layer error, we consider the fact that documents are independent if the query sample is given. Then for any given query sample, we can obtain the following theorem by concentration inequality and symmetrization.

**Theorem 3.** *Denote* $G(S) \triangleq \sup_{f \in \mathcal{F}} (\hat{R}_n(f) - \hat{R}_{n;m_1,\cdots,m_n}(f))$ *and assume* $l \circ \mathcal{F}$ *is bounded by* $M$, *then we have:*

$$P\Big\{G(S) \le \frac{1}{n} \sum_{i=1}^{n} \mathbf{E}_{S_i|q_i}\left[\mathcal{R}_{1;m_i}(l \circ \mathcal{F}(q_i, S_i))\right] + \sqrt{\sum_{i=1}^{n} \frac{2M^2 \log(2/\delta)}{m_i n^2}}\Big|Q\Big\} \le 1 - \delta. \tag{4}$$

*Proof.* First, we prove the bounded difference property for $G(S)$.[2] Given query sample $Q$, all the documents in the document sample will become independent. Denote $S'$ as the document sample obtained by replacing document $z_{j_0}^{i_0}$ in $S$ with a new document $\tilde{z}_{j_0}^{i_0}$. It is clear that

$$\sup_{S,S'} \big|G(S) - G(S')\big| \le \sup_{S,S'} \sup_{f \in \mathcal{F}} \big|\hat{R}_{n;m_1,\cdots,m_n}(f;S) - \hat{R}_{n;m_1,\cdots,m_n}(f;S')\big|$$

$$\le \sup_{S,S'} \sup_{f \in \mathcal{F}} \frac{\sum_{k \ne j_0} \big|(l(z_{j_0}^{i_0}, z_k^{i_0}; f) - l(\tilde{z}_{j_0}^{i_0}, z_k^{i_0}; f)\big|}{n m_{i_0}(m_{i_0} - 1)} \le \frac{2M}{m_{i_0} n}.$$

Then by the McDiarmid's inequality, with probability at least $1 - \delta$, we have

$$G(S) \le \mathbf{E}_{S|Q}[G(S)] + \sqrt{\sum_{i=1}^{n} \frac{2M^2 \log(2/\delta)}{m_i n^2}}. \tag{5}$$

Second, inspired by [9] we introduce permutations to convert the non-sum-of-i.i.d. pairwise loss to a sum-of-i.i.d. form. Assume $\mathcal{S}_{m_i}$ is the symmetric group of degree $m_i$ and $\pi_i \in \mathcal{S}_{m_i}(i = 1, \cdots, n)$ which permutes the $m_i$ documents associates with $q_i$. Since documents associated with the same query follow the identical distribution, we have,

$$\frac{1}{n} \sum_{i=1}^{n} \frac{1}{m_i(m_i - 1)} \sum_{j \ne k} l(z_j^i, z_k^i; f) \stackrel{p}{=} \frac{1}{n} \sum_{i=1}^{n} \frac{1}{m_i!} \sum_{\pi_i} \frac{1}{\lfloor m_i/2 \rfloor} \sum_{j=1}^{\lfloor m_i/2 \rfloor} l(z_{\pi_i(j)}^i, z_{\pi_i(\lfloor m_i/2 \rfloor + j)}^i; f), \tag{6}$$

where $\stackrel{p}{=}$ means identity in distribution. Define a function $\tilde{G}(S_i)$ on each $S_i$ as follows:

$$\tilde{G}(S_i) = \sup_{f \in \mathcal{F}} \Big| \frac{1}{\lfloor m_i/2 \rfloor} \sum_{j=1}^{\lfloor m_i/2 \rfloor} l(z_j^i, z_{\lfloor \frac{m_i}{2} \rfloor + j}^i; f) - \mathbf{E}_{z,z'|q_i}\left[l(z, z'; f)\right] \Big|.$$

We can see that $\tilde{G}(S_i)$ does not contain any document pairs that share a common document. By using Eqn.(6), we can decompose $\mathbf{E}_{S|Q}[G(S)]$ into the sum of $\mathbf{E}_{S_i|q_i}[\tilde{G}(S_i)]$ as below:

$$\mathbf{E}_{S|Q}[G(S)] \le \frac{1}{n} \sum_{i=1}^{n} \frac{1}{m_i!} \sum_{\pi_i} \mathbf{E}_{S_i|q_i}\Big[ \sup_{f \in \mathcal{F}} \Big| \int_{\mathcal{Z}^2} l(z, z'; f) dP(z, z'|q_i)$$

$$- \frac{1}{\lfloor \frac{m_i}{2} \rfloor} \sum_{j=1}^{\lfloor m_i/2 \rfloor} l(Z_{\pi_i(j)}^i, Z_{\pi_i(\lfloor \frac{m_i}{2} \rfloor + j)}^i; f) \Big| \Big] = \frac{1}{n} \sum_{i=1}^{n} \mathbf{E}_{S_i|q_i}\Big[\tilde{G}(S_i)\Big]. \tag{7}$$

Third, we give a bound for $\mathbf{E}_{S_i|q_i}\Big[\tilde{G}(S_i)\Big]$ by use of symmetrization. We introduce a ghost document sample $\tilde{S}_i = \{\tilde{z}_j^i\}_{j=1,\cdots,m_i}$ that is independent of $S_i$ and identically distributed. Assume $\sigma_1^i, \cdots, \sigma_{\lfloor m_i/2 \rfloor}^i$ are independent Rademacher random variables, independent of $S_i$ and $\tilde{S}_i$. Then,

$$\mathbf{E}_{S_i|q_i}\Big[\tilde{G}(S_i)\Big] \le \mathbf{E}_{S_i, \tilde{S}_i|q_i}\left[ \sup_{f \in \mathcal{F}} \Big| \frac{1}{\lfloor m_i/2 \rfloor} \sum_{j=1}^{\lfloor m_i/2 \rfloor} \left( l(z_j^i, z_{\lfloor \frac{m_i}{2} \rfloor + j}^i; f) - l(\tilde{z}_j^i, \tilde{z}_{\lfloor \frac{m_i}{2} \rfloor + j}^i; f) \right) \Big| \right]$$

$$= \mathbf{E}_{S_i, \sigma^i|q_i}\left[ \sup_{f \in \mathcal{F}} \Big| \frac{2}{\lfloor m_i/2 \rfloor} \sum_{j=1}^{\lfloor m_i/2 \rfloor} \sigma_j^i l(z_j^i, z_{\lfloor m_i/2 \rfloor + j}^i; f) \Big| \right] = \mathbf{E}_{S_i|q_i}\left[\mathcal{R}_{1;m_i}(l \circ \mathcal{F}(q_i, S_i))\right]. \tag{8}$$

Jointly considering (5), (7), and (8), we can prove the theorem. ∎

### 4.1.3 Combining the Bounds

Considering Theorem 3 and taking expectation on query sample $Q$, we can obtain that with probability at least $1 - \delta$,

$$\hat{R}_n^l(f) - \hat{R}_{n;m_1,\cdots,m_n}^l(f) \leq \frac{1}{n}\sum_{i=1}^{n}\mathbf{E}_{S_i|q_i}\left[\mathcal{R}_{1;m_i}(l \circ \mathcal{F}(q_i, S_i))\right] + \sqrt{\sum_{i=1}^{n}\frac{2M^2\log\frac{2}{\delta}}{m_i n^2}}, \forall f \in \mathcal{F}.$$

Furthermore, if conventional RA has an upper bound $D(l \circ \mathcal{F}, \cdot)$, for arbitrary sample distribution, document-layer reduced two-layer RA can be upper bounded by $D(l \circ \mathcal{F}, n)$ and query-layer reduced two-layer RA can be bounded by $D(l \circ \mathcal{F}, \lfloor m_i/2 \rfloor)$.

Combining the document-layer error bound and the query-layer error bound presented in the previous subsections, and considering the above discussions, we can eventually prove Theorem 1.

## 4.2 Discussions

According to Theorem 1, we can have the following discussions.

(1) The increasing number of either queries or documents per query in the training data will enhance the two-layer generalization ability. This conclusion seems more intuitive and reasonable than that obtained in [15].

(2) Only if $n \to \infty$ and $m_i \to \infty$ simultaneously does the two-layer generalization bound uniformly converge. That is, if the number of documents for some query is finite, there will always exist document-layer error no matter how many queries have been used for training; if the number of queries is finite, then there will always exist query-layer error, no matter how many documents per query have been used for training.

(3) If we only have a limited budget to label $C$ documents in total, according to Theorem 1, there is an optimal trade off between the number of training queries and that of training documents per query. This is consistent with previous empirical findings in [19]. Actually one can attain the optimal trade off by solving the following optimization problem:

$$\min_{n,m_1,\cdots,m_n} D(l \circ \mathcal{F}, n) + \sqrt{\sum_{i=1}^{n}\frac{2M^2\log\frac{2}{\delta}}{m_i n^2}} + \sum_{i=1}^{n}D(l \circ \mathcal{F}, \lfloor m_i/2 \rfloor)$$

$$s.t. \ \sum_{i=1}^{n}m_i = C$$

This optimum problem is easy to solve. For example, if ranking function class $\mathcal{F}$ satisfies $VC(\tilde{\mathcal{F}}) = V$, for the pairwise 0-1 loss, we have $n^* = \frac{c_1\sqrt{V} + \sqrt{2\log(4/\delta)}}{c_1\sqrt{2V}}\sqrt{C}, m_i^* \equiv \frac{C}{n^*}$ where $c_1$ is a constant. From this result we have the following discussions. (i) $n^*$ decreases with the increasing capacity of the function class. That is, we should label fewer queries and more documents per query when the hypothesis space is larger. (ii) For fixed hypothesis space, $n^*$ increases with the confidence level $\delta$. That is, we should label more query if we want the bound to hold with a larger probability.

The above findings can be used to explain the behavior of existing pairwise ranking algorithms, and can be used to guide the construction of training set for learning to rank.

## 5 Conclusions and Discussions

In this paper, we have proposed conducting two-layer generalization analysis for ranking, and proved a two-layer generalization bound for ERM learning with pairwise losses. The theoretical results we have obtained can better explain experimental observations in learning to rank than previous results, and can provide general guidelines to trade off between deep labeling and shallow labeling in the construction of training data.

For future work, we plan to i) extend our analysis to listwise loss functions in ranking, such as ListNet [7] and listMLE [18]; ii) and introduce noise condition in order to obtain faster convergency.

## Footnotes

[1]In [15], although a similar sampling strategy to the two-layer sampling is mentioned, the generalization analysis, however, does not consider the independent sampling at the document layer. As a result, the generalization bound they obtained is a query-level generalization bound, but not a two-layer generalization bound.

[2] We say a function has bounded difference, if the value of the function can only have bounded change when only one variable is changed.

# References

[1] S. Agarwal, T. Graepel, R. Herbrich, S.Har-Peled, and D. Roth. Generalization bounds for the area under the roc curve. *Journal of Machine Learning Research*, 6:393–425, 2005.

[2] S. Agarwal and P. Niyogi. Generalization bounds for ranking algorithms via algorithmic stability. *Journal of Machine Learning Research*, 10:441–474, 2009.

[3] P. L. Bartlett, S. Mendelson, and M. Long. Rademacher and gaussian complexities: risk bounds and structural results. *Journal of Machine Learning Research*, 3:463–482, 2002.

[4] J. Baxter. Learning internal representations. In *Proceedings of the Eighth International Conference on Computational Learning Theory*, pages 311–320. ACM Press, 1995.

[5] C. Burges, T. Shaked, E. Renshaw, A. Lazier, M. Deeds, N. Hamilton, and G. Hullender. Learning to rank using gradient descent. In *ICML '05: Proceedings of the 22nd International Conference on Machine learning*, pages 89–96, 2005.

[6] Y. Cao, J. Xu, T. Y. Liu, H. Li, Y. Huang, and H. W. Hon. Adapting ranking svm to document retrieval. In *SIGIR '06: Proceedings of the 29th Annual International ACM SIGIR Conference on Research and Development in Information Retrieval*, pages 186–193. ACM Press, 2006.

[7] Z. Cao, T. Qin, T. Y. Liu, M. F. Tsai, and H. Li. Learning to rank: from pairwise approach to listwise approach. In *ICML '07: Proceedings of the 24th International Conference on Machine learning*, pages 129–136, 2007.

[8] C. L. Clarke, N. Craswell, and I. Soboroff. Overview of the trec 2009 web track. Technical report, no date.

[9] S. Clémençon, G. Lugosi, and N. Vayatis. Ranking and scoring using empirical risk minimization. In *COLT '05: Proceedings of the 18th Annual Conference on Learning Theory*, pages 1–15, 2005.

[10] D. Cossock and T. Zhang. Subset ranking using regression. In *COLT '06: Proceedings of the 19th Annual Conference on Learning Theory*, pages 605–619, 2006.

[11] Y. Freund, R. Iyer, R. E. Schapire, and Y. Singer. An efficient boosting algorithm for combining preferences. *Journal of Machine Learning Research*, 4:933–969, 2003.

[12] R. Herbrich, T. Graepel, and K. Obermayer. Large margin rank boundaries for ordinal regression. In *Advances in Large Margin Classifiers*, pages 115–132, Cambridge, MA, 1999. MIT.

[13] T. Joachims. Optimizing search engines using clickthrough data. In *KDD '02: Proceedings of the 8th ACM SIGKDD international conference on Knowledge discovery and data mining*, pages 133–142, 2002.

[14] Y. Y. Lan, T. Y. Liu, Z. M. Ma, and H. Li. Generalization analysis of listwise learning-to-rank algorithms. In *ICML '09: Proceedings of the 26th International Conference on Machine Learning*, pages 577–584, 2009.

[15] Y. Y. Lan, T. Y. Liu, T. Qin, Z. M. Ma, and H. Li. Query-level stability and generalization in learning to rank. In *ICML '08: Proceedings of the 25th International Conference on Machine Learning*, pages 512–519, 2008.

[16] T. Y. Liu. Learning to rank for information retrieval. *Foundations and Trends in Information Retrieval*, 3:225–331, 2009.

[17] J. R. Wen, J. Y. Nie, and H. J. Zhang. Clustering user queries of a search engine. In *WWW '01: Proceedings of the 10th international conference on World Wide Web*, pages 162–168, New York, NY, USA, 2001. ACM.

[18] F. Xia, T.-Y. Liu, J. Wang, W. Zhang, and H. Li. Listwise approach to learning to rank - theory and algorithm. In *ICML '08: Proceedings of the 25th International Conference on Machine learning*, pages 1192–1199. Omnipress, 2008.

[19] E. Yilmaz and S. Robertson. Deep versus shallow judgments in learning to rank. In *SIGIR '09: Proceedings of the 32th annual international ACM SIGIR conference on Research and development in information retrieval*, pages 662–663, 2009.

